# Neurobiology, Psychophysics, and Computational Models of Visual Attention

**Ernst Niebur**
Computation and Neural Systems
California Institute of Technology
Pasadena, CA 91125, USA

**Bruno A. Olshausen**
Department of Anatomy and Neurobiology
Washington University School of Medicine
St. Louis, MO 63110

The purpose of this workshop was to discuss both recent experimental findings and computational models of the neurobiological implementation of selective attention. Recent experimental results were presented in two of the four presentations given (C.E. Connor, Washington University and B.C. Motter, SUNY and V.A. Medical Center, Syracuse), while the other two talks were devoted to computational models (E. Niebur, Caltech, and B. Olshausen, Washington University).

Connor presented the results of an experiment in which the receptive field profiles of V4 neurons were mapped during different states of attention in an awake, behaving monkey. The attentional focus was manipulated in this experiment by altering the position of a behaviorally relevant ring-shaped stimulus. The animal's task was to judge the size of the ring when compared to a reference ring (i.e., same or different). In order to map the receptive field profile, a behaviorally irrelevant bar stimulus was flashed at one of several positions inside and outside the classical receptive field (CRF). It was found that shifts of attention produced alterations in receptive field profiles for over half the cells studied. In most cases the receptive field center of gravity translated towards attentional foci in or near the CRF. Changes in width and shape of the receptive field profile were also observed, but responsive regions were not typically limited to the location of the attended ring stimulus. Attention-related effects often included enhanced responses at certain locations as well as diminished responses at other locations.

Motter studied the basic mechanisms of visual search as manifested in the single unit activity of rhesus monkeys. The animals were trained to select a bar stimulus among others based on the color or luminance of the target stimulus. The majority of neurons were selectively activated when the color or luminance of the stimulus in the receptive field matched the color or luminance of the cue, whereas the activity was attenuated when there was no match. Since a cell responds differently to the same stimulus depending on the color or luminance of the cue (which is given far away from the stimulus by the color or luminance of the fixation spot), the activity of the neurons reflect a selection based on the cued feature and not simply the physical color or luminance of the receptive field stimulus. Motter showed that the

selection can also be based on memory by switching off the cue in the course of the experiment. The monkey could then perform the task only by relying on his memory and the pattern of V4 activity. In the memory-based task as well as in the experiments with the stimulus continuously present, the differential activation was independent of spatial location and offers therefore a physiological correlate to psychophysical studies suggesting that stimuli can be preferentially selected in parallel across the visual field.

Niebur presented a model for the neuronal implementation of selective visual attention based on temporal correlation among groups of neurons. In the model, neurons in primary visual cortex respond to visual stimuli with a Poisson distributed spike train with an appropriate, stimulus-dependent mean firing rate. The spike trains of neurons whose receptive fields do *not* overlap with the "focus of attention" are distributed according to homogeneous (time-independent) Poisson process with no correlation between action potentials of different neurons. In contrast, spike trains of neurons with receptive fields within the focus of attention are distributed according to non-homogeneous (time-dependent) Poisson processes. Since the short-term average spike rates of all neurons with receptive fields in the focus of attention covary, correlations between these spike trains are introduced which are detected by inhibitory interneurons in V4. These cells, modeled as modified integrate-and-fire neurons, function as coincidence detectors and suppress the response of V4 cells associated with non-attended visual stimuli. The model reproduces quantitatively experimental data obtained in cortical area V4 of monkey.

The model presented by Olshausen proposed that attentional gating takes place via an explicit control process, without relying on temporal correlation. This model is designed to serve as a possible explanation for how the visual cortex forms position and scale invariant representations of objects. Control neurons dynamically modify the synaptic strengths of intracortical connections so that information from a windowed region of primary visual cortex is selectively routed to higher cortical areas, preserving spatial relationships. The control signals for setting the position and size of the attentional window are hypothesized to originate from neurons in the pulvinar and in the deep layers of visual cortex. The dynamics of these control neurons are governed by simple differential equations that can be realized by neurobiologically plausible circuits. In pre-attentive mode, the control neurons receive their input from a low-level "saliency map" representing potentially interesting regions of a scene. During the pattern recognition phase, control neurons are driven by the interaction between top-down (memory) and bottom-up (retinal input) sources. The model predicts that the receptive fields of cortical neurons should shift with attention, as found in Connor's experiments, although the predicted shifts are somewhat larger than those found to date.

## Acknowledgement

The work of EN and BAO was supported by the Office of Naval Research. EN was additionally supported by the National Science Foundation.